# Neural mechanisms of contrast dependent receptive field size in V1

**Jim Wielaard and Paul Sajda**
Department of Biomedical Engineering
Columbia University
New York, NY 10027
(djw21, ps629)@columbia.edu

## Abstract

Based on a large scale spiking neuron model of the input layers $4C\alpha$ and $\beta$ of macaque, we identify neural mechanisms for the observed contrast dependent receptive field size of V1 cells. We observe a rich variety of mechanisms for the phenomenon and analyze them based on the relative gain of excitatory and inhibitory synaptic inputs. We observe an average growth in the spatial extent of excitation and inhibition for low contrast, as predicted from phenomenological models. However, contrary to phenomenological models, our simulation results suggest this is neither sufficient nor necessary to explain the phenomenon.

## 1 Introduction

Neurons in the primary visual cortex (V1) display what is often referred to as "size tuning", i.e. the response of a cell is maximal around a cell-specific stimulus size and generally decreases substantially (30-40% on average) or vanishes altogether for larger stimulus sizes[1−9]. The cell-specific stimulus size eliciting a maximum response, also known as the "receptive field size" of the cell[4], has a remarkable property in that it is not contrast invariant, unlike for instance orientation tuning in V1. Quite the contrary, the contrast-dependent change in receptive field size of V1 cells is profound. Typical is a doubling in receptive field size for stimulus contrasts decreasing by a factor of 2-3 on the linear part of the contrast response function[4]. This behavior is seen throughout V1, including all cell types in all layers and at all eccentricities. A functional interpretation of the phenomenon is that neurons in V1 sacrifice spatial resolution in return for a gain in contrast sensitivity at low contrasts [4]. However, its neural mechanisms are at present very poorly understood. Understanding these mechanisms is potentially important for developing a theoretical model of early signal integration and neural encoding of visual features in V1.

We have recently developed a large-scale spiking neuron model that accounts for the phenomenon and suggests neural mechanisms from which it may originate. This paper provides a technical description of these mechanisms.

## 2 The model

Our model consists of 8 ocular dominance columns and 64 orientation hypercolumns (i.e. pinwheels), representing a 16 $mm^2$ area of a macaque V1 input layer $4C\alpha$ or $4C\beta$. The

model consists of approximately 65,000 cortical cells in each of the four configurations (see below), and the corresponding appropriate number of LGN cells. Our cortical cells are modeled as conductance based integrate-and-fire point neurons, 75% are excitatory cells and 25% are inhibitory cells. Our LGN cells are rectified spatio-temporal linear filters. The model is constructed with isotropic short-range cortical connections ($< 500\mu m$), realistic LGN receptive field sizes and densities, realistic sizes of LGN axons in V1, and cortical magnification factors and receptive field scatter that are in agreement with experimental observations.

Dynamic variables of a cortical model-cell $i$ are its membrane potential $v_i(t)$ and its spike train $\mathcal{S}_i(t) = \sum_k \delta(t - t_{i,k})$, where $t$ is time and $t_{i,k}$ is its $k$th spike time. Membrane potential and spike train of each cell obey a set of $N$ equations of the form

$$C_i \frac{dv_i}{dt} = -g_{L,i}(v_i - v_L) - g_{E,i}(t, [\mathcal{S}]_E, \eta_E)(v_i - v_E)$$

$$-g_{I,i}(t, [\mathcal{S}]_I, \eta_I)(v_i - v_I) \, , \; i = 1, \ldots, N \, . \tag{1}$$

These equations are integrated numerically using a second order Runge-Kutta method with time step 0.1 ms. Whenever the membrane potential reaches a fixed threshold level $v_T$ it is reset to a fixed reset level $v_R$ and a spike is registered. The equation can be rescaled so that $v_i(t)$ is dimensionless and $C_i = 1$, $v_L = 0$, $v_E = 14/3$, $v_I = -2/3$, $v_T = 1$, $v_R = 0$, and conductances (and currents) have dimension of inverse time.

The quantities $g_{E,i}(t, [\mathcal{S}], \eta_E)$ and $g_{I,i}(t, [\mathcal{S}], \eta_I)$ are the excitatory and inhibitory conductances of neuron $i$. They are defined by interactions with the other cells in the network, external noise $\eta_{E(I)}$, and, in the case of $g_{E,i}$ possibly by LGN input. The notation $[\mathcal{S}]_{E(I)}$ stands for the spike trains of all excitatory (inhibitory) cells connected to cell $i$. Both, the excitatory and inhibitory populations consist of two subpopulations $\mathcal{P}_k(E)$ and $\mathcal{P}_k(I)$, $k = 0, 1$, a population that receives LGN input ($k = 1$) and one that does not ($k = 0$). In the model presented here 30% of both the excitatory and inhibitory cell populations receive LGN input. We assume noise, cortical interactions and LGN input act additively in contributing to the total conductance of a cell,

$$g_{E,i}(t, [\mathcal{S}]_E, \eta_E) = \eta_{E,i}(t) + g^{cor}_{E,i}(t, [\mathcal{S}]_E) + \delta_i g^{LGN}_i(t)$$

$$g_{I,i}(t, [\mathcal{S}]_I, \eta_I) = \eta_{I,i}(t) + g^{cor}_{I,i}(t, [\mathcal{S}]_I) \, , \tag{2}$$

where $\delta_i = \ell$ for $i \in \{\mathcal{P}_\ell(E), \mathcal{P}_\ell(I)\}$, $\ell = 0, 1$. The terms $g^{cor}_{\mu,i}(t, [\mathcal{S}]_\mu)$ are the contributions from the cortical excitatory ($\mu = E$) and inhibitory ($\mu = I$) neurons and include only isotropic connections,

$$g^{cor}_{\mu,i}(t, [\mathcal{S}]_\mu) =$$

$$\int_{-\infty}^{+\infty} ds \sum_{k=0}^{1} \sum_{j \in \mathcal{P}_k(\mu)} \mathcal{C}^{k',k}_{\mu',\mu}(||\vec{x}_i - \vec{x}_j||) G_{\mu,j}(t - s) \mathcal{S}_j(s) \, , \tag{3}$$

where $i \in \mathcal{P}_{k'}(\mu')$ Here $\vec{x}_i$ is the spatial position (in cortex) of neuron $i$, the functions $G_{\mu,j}(\tau)$ describe the synaptic dynamics of cortical synapses and the functions $\mathcal{C}^{k',k}_{\mu',\mu}(r)$ describe the cortical spatial couplings (cortical connections). The length scale of excitatory and inhibitory connections is about $200\mu$m and $100\mu$m respectively.

In agreement with experimental findings (see references in [10]), the LGN neurons are modeled as rectified center-surround linear spatiotemporal filters. The LGN temporal kernels are modeled in agreement with [11], and the LGN spatial kernels are of center-surround type.

An important class of parameters are those that define and relate the model's geometry in visual space and cortical space. Geometric properties are different for the two input layers $4C\alpha, \beta$ and depend also on the eccentricity. As said, contrast dependent receptive field size is observed to be insensitive to those differences[4−6,8]. In order to verify that our explanations are consistent with this observation, we have performed numerical simulations for four different sets of parameters, corresponding to the $4C\alpha, \beta$ layers at para-foveal eccentricities ($< 5°$) and at eccentricities around $10°$. These different model configurations are referred to as M0, M10, and P0, P10. Reported results are qualitatively similar for all four configurations unless otherwise noted. The above is only a very brief description of the model, the details can be found in [12].

## 3  Visual stimuli and data collection

The stimulus used to analyze the phenomenon is a drifting grating confined to a circular aperture, surrounded by a blank (mean luminance) background. The luminance of the stimulus is given by $I(\vec{y}, t) = I_0(1 + \epsilon \cos(\omega t - \vec{k} \cdot \vec{y} + \phi))$ for $||\vec{y}|| \leq r_A$ and $I(\vec{y}, t) = I_0$ for $||\vec{y}|| > r_A$, with average luminance $I_0$, contrast $\epsilon$, temporal frequency $\omega$, spatial wave vector $\vec{k}$, phase $\phi$, and aperture radius $r_A$. The aperture is centered on the receptive field of the cell and varied in size, while the other parameters are kept fixed and set to preferred values. All stimuli are presented monocularly. Samples consisting of approximately 200 cells were collected for each configuration, containing about an equal number of simple and complex cells. The experiments were performed at "high" contrast, $\epsilon = 1$, and "low" contrast, $\epsilon = 0.3$.

## 4  Approximate model equations

We find that, to good approximation, the membrane potential and instantaneous firing rate of our model cells are respectively[12,13]

$$\langle v_k(t, r_A) \rangle \approx V_k(r_A, t) \equiv \frac{\langle I_{D,k}(t, r_A) \rangle}{\langle g_{T,k}(t, r_A) \rangle} \, , \tag{4}$$

$$\langle \mathcal{S}_k(t, r_A) \rangle \approx f_k(t, r_A) \equiv \delta_k \left[ \langle I_{D,k}(t, r_A) \rangle - \langle g_{T,k}(t, r_A) \rangle - \Delta_k \right]_+ \, , \tag{5}$$

where $[x]_+ = x$ if $x \geq 0$ and $[x]_+ = 0$ if $x \leq 0$, and where, the gain $\delta_k$ and threshold $\Delta_k$ do not depend on the aperture radius $r_A$ for most cells. The total conductance $g_{T,k}(t, r_A)$ and difference current $I_{D,k}(t, r_A)$ are given by

$$g_{T,k}(t, r_A) = g_L + g_{E,k}(t, [\mathcal{S}]_E, r_A) + g_{I,k}(t, [\mathcal{S}]_I, r_A) \tag{6}$$

$$I_{D,k}(r_A, t) = g_{E,k}(t, [\mathcal{S}]_E, r_A) \, V_E - g_{I,k}(t, [\mathcal{S}]_I, r_A) \, |V_I| \, . \tag{7}$$

## 5  Mechanisms of contrast dependent receptive field size

From Eq. (4) and (5) it follows that a change in receptive field size in general results from a change in behavior of the relative gain,

$$G(r_A) = \frac{\partial g_E / \partial r_A}{\partial g_I / \partial r_A} \, . \tag{8}$$

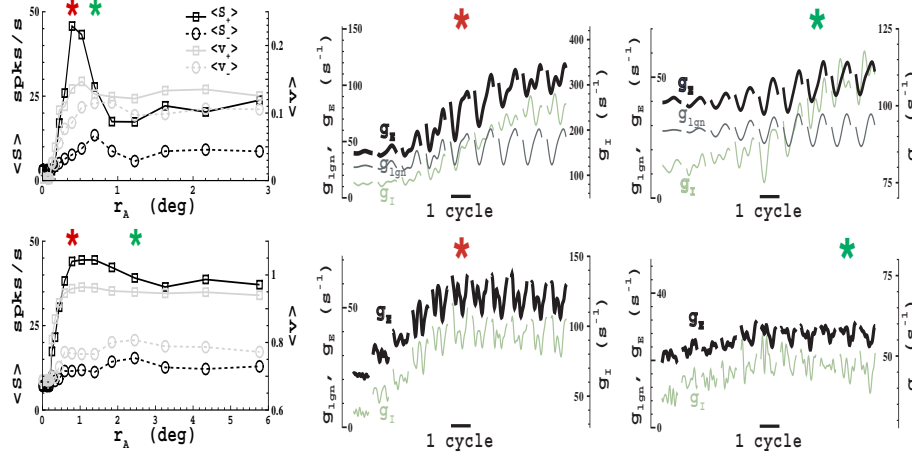

Figure 1: Two example cells, an M0 simple cell which receives LGN input (top) and an M10 complex cell which does not (bottom). (column 1) Responses as function of aperture size. Mean responses are plotted for the complex cell, first harmonic for the simple cell. Apertures of maximum of responses (i.e. receptive field sizes) are indicated with asterisks (dark=high contrast, light=low contrast). (column 2) Conductances for high contrast at apertures near the maximum responses. Conductances are displayed as nine (top) and eleven (bottom) sub-panels giving the cycle-trial averaged conductances as a function of time (relative to cycle) and aperture size. (column 3) Conductances for low contrast at apertures near the maximum responses. Asterisks in the conductance figures (columns 2 and 3) indicate corresponding apertures of maximum response (column 1)

Note that this is a rather different parameter than the "surround gain" parameter ($k_s$) used in the ratio-of-Gaussians (ROG) model[8]–e.g. unlike for $k_s$, there is no one-to-one relationship between $G(r_A)$ and the degree of surround suppression. Qualitatively, the conductances show a similar dependence on aperture size as the membrane potential responses and spike responses, i.e. they display surround suppression as well [12]. Receptive field sizes based on these conductances are a measure of the spatial summation extent of excitation and inhibition.

An obvious way to change the behavior of $G$, and consequently the receptive field size, is to change the spatial summation extent of $g_E$ and/or $g_I$. However this is not strictly necessary. For example, other possibilities are illustrated by the two cells in Fig. 1. These cells show, both in spike and membrane potential responses, a receptive field growth of a factor of 2 (top) and 3 (bottom) at low contrast. However, for both cells the spatial summation extent of excitation at low contrast is one aperture less than at high contrast.

In a similar way as for spike train responses, we also obtained receptive field sizes for the conductances. As do spike responses (Fig. 2A), both excitation and inhibition (Fig. 2B&C) also show, on the average, an increase in their spatial summation extent as contrast is decreased, but the increase is in general smaller than what is seen for spike responses, particularly for cells that show significant receptive field growth. For instance, we see from Figure 2B and C that for cells in the sample with receptive field growths $\sim 2$ or greater, the growth for the conductances is always considerably less than the growth based on spike responses. Expressed more rigorously, a Wilcoxon test on ratio of growth ratios larger than unity gives $p < 0.05$ (all cells, excitation, Fig. 2B), $p < 0.15$ (all cells, inhibition, Fig. 2C), $p < 0.001$ (cells with receptive field growth rate $r_+/r_- > 1.5$, both excitation and inhibition.) Although some increase in the spatial summation extent of excitation and

inhibition is in general the rule, this increase is rather arbitrary and bears not much relation with the receptive field growth based on spike responses. The same conclusions follow from membrane potential responses (not shown).

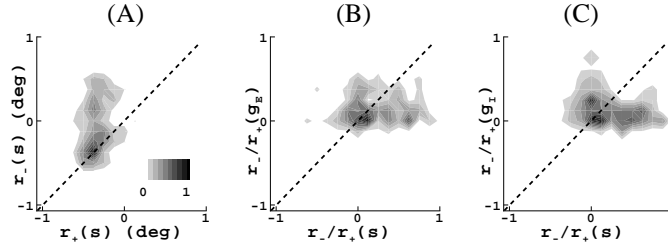

Figure 2: (A) Joint distribution of high and low contrast receptive field sizes, $r_+$ and $r_-$, based on spike responses. All scales are logarithmic, base 10. All distributions are normalized to a peak value of one. Receptive field growth at low contrast is clear. Average growth ratio is 1.9 and is significantly greater than unity (Wilcoxon test, $p < 0.001$). (B & C) Joint distributions of receptive field growth and growth of spatial summation extent of excitation (B) and inhibition (C) (computed as ratios). There is no simple relation between receptive field growth and the growth of the spatial summation extent of excitatory or inhibitory inputs. For cells in the sample with larger receptive field growths (factor of $\sim 2$ or greater) this growth is always considerably larger than the growths of their excitatory and inhibitory inputs.

Fig. 2 thus demonstrates that, contrary to what is predicted by the difference-of-Gaussians (DOG) [4] and ROG models [8] (see Discussion), a growth of spatial summation extent of excitation (and/or inhibition) at low contrast is neither sufficient nor necessary to explain the receptive field growth seen in spike responses. Membrane potential responses give the same conclusion. The fact that a change in receptive field size can take place without a change in the spatial summation extent of $g_E$ or $g_I$ can be illustrated by a simple example.

Consider a situation where both $g_E$ and $g_I$ have their maximum at the same aperture size $r_E = r_I = r_\star$ and are monotonically increasing for $r_A < r_\star$ and monotonically decreasing for $r_A > r_\star$, as depicted in Fig. 3. We can distinguish three classes with respect to the relative location of the maxima in spike responses $r_S$ and the conductances $r_\star$, namely {X: $r_S < r_\star$}, {Y: $r_S = r_\star$} and {Z: $r_S > r_\star$}. It follows from (5) that if we define the

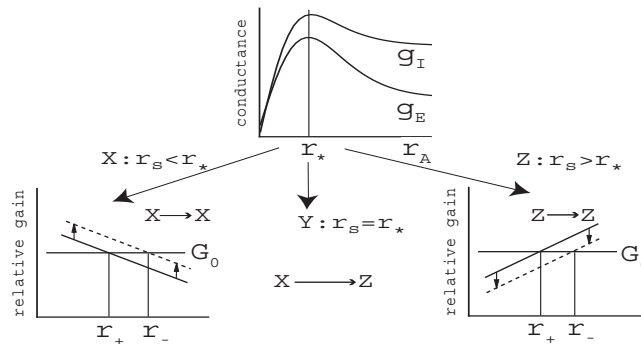

Figure 3: Schematic illustration of mechanisms for receptive field growth under equal and constant spatial summation extent of the conductances ($r_E = r_I = r_\star$).

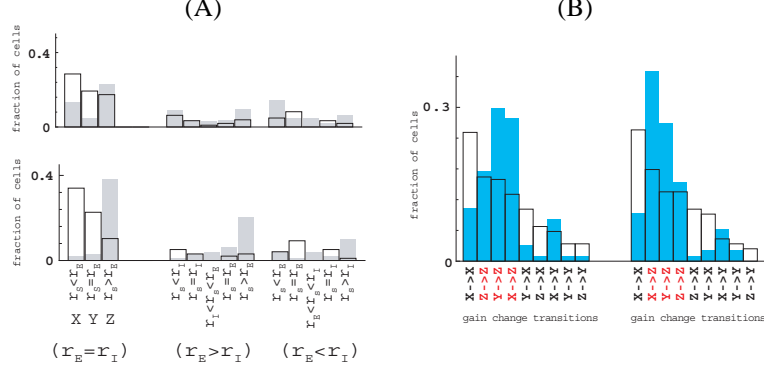

Figure 4: (A) Distributions of the relative positions of the maxima (receptive field sizes) of spike responses $r_S$ and conductances $r_E$ and $r_I$, for the M0 configuration. A division is made with respect to the maxima in the conductances, this corresponds to the left ($r_E = r_I$), central ($r_E > r_I$), and right ($r_E < r_I$) part of the figure. Each panel is further subdivided with respect to the maximum in the spike response $r_S$. Upper histograms are for all cells in the sample, lower histograms are for cells that have receptive field growth $r_-/r_+ > 1.5$. Unfilled histograms are for high contrast, shaded histograms are for low contrast. (B) Prevalence of transitions between positions of maxima in spike responses and excitatory conductances (left) and in spike responses and inhibitory conductances (right) for a high $\rightarrow$ low contrast change. See text for definitions of X, Y, Z classes. Data are evaluated for all cells (unfilled histograms) and for cells with a receptive field growth $r_-/r_+ > 1.5$ (shaded histograms).

parameter $G_0(v) = (|v_I| + v)/(v_E - v)$ then we can characterize the difference between classes X and Z by the way that $G$ crosses $G_0(1)$ around $r_S$ as depicted in Fig. 3. For class Y the parameter G is not of any particular interest as it can assume arbitrary behavior around $r_S$. It follows from (4) that similar observations hold for the maximum in the membrane potential $r_v$ and we need simply to replace $G_0(1)$ with $G_0(v(r_v))$. A growth of receptive field size can occur without any change in the spatial summation extent ($r_\star$) of the conductances. Suppose we wish to remain within the same class X or Z, then receptive field growth, can be induced, for instance, by an overall increase (X) or an overall decrease (Z) in relative gain $G(r_A)$ as shown in Fig. 3 (dashed line). Receptive field growth also can be caused by more drastic changes in $G$ so that the transitions X $\rightarrow$ Y, X $\rightarrow$ Z or Y $\rightarrow$ Z occur for a high $\rightarrow$ low contrast change The situation is somewhat more involved when we allow for non-suppressed responses and conductances, and for different positions of the maxima of $g_E$ and $g_I$, however, the essence of our conclusions remains the same.

Analysis of our data in the light of the above example is given in Fig. 4. Cells were classified (Fig. 4A) according to the relative positions of their maxima in spike response ($r_S$) and excitatory ($r_E$) and inhibitory ($r_I$) conductances, using F0+F1 (i.e. mean response + first Fourier component of the response). Membrane potential responses yield similar results. Comparing this classification at high and low contrast we observe a striking difference for cells with significant receptive field growths, i.e. with growth ratios >1.5 (Fig. 4A, bottom), indicative of X $\rightarrow$ Y, X $\rightarrow$ Z and Y $\rightarrow$ Z transitions (as discussed in the simplified example above). In this realistic situation there are of course many more transitions (i.e. $13^2$), however, that we indeed observe a prevalence for these transitions can be demonstrated in two ways using slightly modified definitions of the X,Y,Z classes. First (Fig. 4B, left), if we redefine the X,Y,Z classes with respect to $r_S$ and $r_E$ while ignoring $r_I$, i.e. {X: $r_S < r_E$}, {Y: $r_S = r_E$} and {Z: $r_S > r_E$}, then the transition distribution for cells with significant receptive field growth shows that in about 60% of these cells a X $\rightarrow$ Z or

$Y \rightarrow Z$ transition occurs. Taken together with the fact that roughly 10% of the cells with significant receptive field growth (Figure 4A, bottom) have $r_I \leq r_S < r_E$ at high contrast and $r_E < r_S \leq r_I$ at low contrast, we can conclude that for more than 50% of the cells with significant receptive field growth, a transition takes place from a high contrast RF size less or equal to the spatial summation extent of excitation and inhibition, to a low contrast receptive field size which exceeds both (by at least one aperture). Note that these transitions occur in addition to any growth of $r_E$ or $r_I$. Secondly (Fig. 4B, right), the same conclusion is reached when we redefine the X,Y,Z classes with respect to $r_S$ and $r_I$ while ignoring $r_E$ ({X: $r_S < r_I$}, {Y: $r_S = r_I$} and {Z: $r_S > r_I$}), Now a $X \rightarrow Z$ or $Y \rightarrow Z$ transition occurs in about 70% of the cells with significant receptive field growth, while about 20% of the cells with significant receptive field growth (Fig. 4A, bottom) have $r_E \leq r_S < r_I$ at high contrast and $r_I < r_S \leq r_E$ at low contrast. Finally, Fig. 4B also demonstrates the presence of a rich diversity in relative gain changes in our model, since all transitions (for all cells, unfilled histograms) occur with some reasonable probability.

## 6    Discussion

The DOG model suggests that growth in receptive field size at low contrast is due to an increase of the spatial summation extent of excitation[4] (i.e. increase in the spatial extent parameter $\sigma_E$). This was partially confirmed experimentally in cat primary visual cortex[7]. Although it has been claimed[8] that the ROG model could explain receptive field growth solely from a change in the relative gain parameter $k_s$, we believe this is incorrect. Since there is a one-to-one relationship between $k_s$ and surround suppression, this would imply that contrast dependent receptive field size simply results from contrast dependent surround suppression, which contradicts experimental data[4,8]. As does the DOG model, the ROG model, based on analysis of our data, also predicts that contrast dependent receptive field size is due to contrast dependence of the spatial summation extent of excitation. As we have shown, our simulations confirm an average growth of spatial summation extent of excitation (and inhibition) at low contrast. However, this growth is neither sufficient nor necessary to explain receptive field growth. For cells with significant receptive field growth, ($r_+/r_- > 1.5$) we were able to identify an additional property of the neural mechanisms. For more than 50% of such cells, a transition takes place from a high contrast RF size less or equal to the spatial summation extent of excitation and inhibition, to a low contrast receptive field size which exceeds both.

An important characteristic of our model is that it is not specifically designed to produce the phenomenon. Rather, the model parameters are set such that it produces realistic orientation tuning and a realistic distribution of response modulations in response to drifting gratings (simple & complex cells). Constructed in this way, our model then naturally produces a wide variety of realistic response properties, classical as well as extraclassical, including the phenomenon discussed here. A prominent feature of the mechanisms we suggest is that, contrary to common belief, they require neither the long-range lateral connections in V1 [14−18] nor extrastriate feedback [6,8,19,20]. The average receptive field growth we see in our model is about a factor of two ($r_-/r_+ \sim 2$). This is a little less than what is observed in experiments [5,8]. This leaves room for contributions from the LGN input. It seems reasonable to assume that contrast dependent receptive field size is not limited to V1 and is also a property of LGN cells. Somewhat surprisingly, this has to our knowledge not been verified yet for macaque. Contrast dependent receptive field size of LGN cells has been observed in marmoset and an average growth ratio at low contrast of 1.3 was reported[21]. Receptive field growth of LGN cells in some sense introduces an overall geometric scaling

factor on the entire visual input to V1. This observation ignores a great many details of course. For instance, the fact that the density of LGN cells (LGN receptive fields) is not known to change with contrast. On the other hand, it seems unlikely that a reasonable receptive field expansion of LGN cells would not be at least partially transferred to V1. Thus it seems reasonable to conclude from our work that the phenomenon in V1, in particular that seen in layer 4, may be attributed largely to isotropic short-range ($< 0.5$ mm) cortical connections and LGN input.

## Acknowledgments

This work was supported by grants from ONR (MURI program, N00014-01-1-0625) and NGA (HM1582-05-C-0008).

## References

[1] Dow, B, Snyder, A, Vautin, R, & Bauer, R. (1981) *Exp Brain Res* **44**, 213–228.

[2] Schiller, P, Finlay, B, & Volman, S. (1976) *J Neurophysiol* **39**, 1288–1319.

[3] Silito, A, Grieve, K, Jones, H, Cudeiro, J, & Davis, J. (1995) *Nature* **378**, 492–496.

[4] Sceniak, M, Ringach, D, Hawken, M, & Shapley, R. (1999) *Nat Neurosci* **2**, 733–739.

[5] Kapadia, M, Westheimer, G, & Gilbert, C. (1999) *Proc Nat Acad Sci USA* **96**, 12073–12078.

[6] Sceniak, M, Hawken, M, & Shapley, R. (2001) *J Neurophysiol* **85**, 1873–1887.

[7] Anderson, J, Lampl, I, Gillespie, D, & Ferster, D. (2001) *J Neurosci* **21**, 2104–2112.

[8] Cavanaugh, J, Bair, W, & Movshon, J. (2002) *J Neurophysiol* **88**, 2530–2546.

[9] Ozeki, H, Sadakane, O, Akasaki, T, Naito, T, Shimegi, S, & Sato, H. (2004) *J Neurosci* **24**, 1428–1438.

[10] McLaughlin, D, Shapley, R, Shelley, M, & Wielaard, J. (2000) *Proc Nat Acad Sci USA* **97**, 8087–8092.

[11] Benardete, E & Kaplan, E. (1999) *Vis Neurosci* **16**, 355–368.

[12] Wielaard, J & Sajda, P. (2005) *Cerebral Cortex* in press.

[13] Wielaard, J, Shelley, M, McLaughlin, D, & Shapley, R. (2001) *J Neurosci* **21(14)**, 5203–5211.

[14] DeAngelis, G, Freeman, R, & Ohzawa, I. (1994) *J Neurophysiol* **71**, 347–374.

[15] Somers, D, Todorov, E, Siapas, A, Toth, L, Kim, D, & Sur, M. (1998) *Cereb Cortex* **8**, 204–217.

[16] Dragoi, V & Sur, M. (2000) *J Neurophysiol* **83**, 1019–1030.

[17] Hupé, J, James, A, Girard, P, & Bullier, J. (2001) *J Neurophysiol* **85**, 146–163.

[18] Stettler, D, Das, A, Bennett, J, & Gilbert, C. (2002) *Neuron* **36**, 739–750.

[19] Angelucci, A, Levitt, J, Walton, E, Hupé, J, Bullier, J, & Lund, J. (2002) *J Neurosci* **22**, 8633–8646.

[20] Bair, W, Cavanaugh, J, & Movshon, J. (2003) *J Neurosci* **23(20)**, 7690–7701.

[21] Solomon, S, White, A, & Martin, P. (2002) *J Neurosci* **22(1)**, 338–349.
